# Reinforcement Learning Methods for Continuous-Time Markov Decision Problems

**Steven J. Bradtke**
Computer Science Department
University of Massachusetts
Amherst, MA 01003
bradtke@cs.umass.edu

**Michael O. Duff**
Computer Science Department
University of Massachusetts
Amherst, MA 01003
duff@cs.umass.edu

## Abstract

Semi-Markov Decision Problems are continuous time generalizations of discrete time Markov Decision Problems. A number of reinforcement learning algorithms have been developed recently for the solution of Markov Decision Problems, based on the ideas of asynchronous dynamic programming and stochastic approximation. Among these are TD($\lambda$), $Q$-learning, and Real-time Dynamic Programming. After reviewing semi-Markov Decision Problems and Bellman's optimality equation in that context, we propose algorithms similar to those named above, adapted to the solution of semi-Markov Decision Problems. We demonstrate these algorithms by applying them to the problem of determining the optimal control for a simple queueing system. We conclude with a discussion of circumstances under which these algorithms may be usefully applied.

## 1 Introduction

A number of reinforcement learning algorithms based on the ideas of asynchronous dynamic programming and stochastic approximation have been developed recently for the solution of Markov Decision Problems. Among these are Sutton's TD($\lambda$) [10], Watkins' $Q$-learning [12], and Real-time Dynamic Programming (RTDP) [1,

3]. These learning alogorithms are widely used, but their domain of application has been limited to processes modeled by discrete-time Markov Decision Problems (MDP's).

This paper derives analogous algorithms for semi-Markov Decision Problems (SMDP's) — extending the domain of applicability to continuous time. This effort was originally motivated by the desire to apply reinforcement learning methods to problems of adaptive control of queueing systems, and to the problem of adaptive routing in computer networks in particular. We apply the new algorithms to the well-known problem of routing to two heterogeneous servers [7]. We conclude with a discussion of circumstances under which these algorithms may be usefully applied.

## 2   Semi-Markov Decision Problems

A semi-Markov process is a continuous time dynamic system consisting of a countable state set, $\mathcal{X}$, and a finite action set, $\mathcal{A}$. Suppose that the system is originally observed to be in state $x \in \mathcal{X}$, and that action $a \in \mathcal{A}$ is applied. A semi-Markov process [9] then evolves as follows:

- The next state, $y$, is chosen according to the transition probabilities $P_{xy}(a)$

- A reward rate $\rho(x, a)$ is defined until the next transition occurs

- Conditional on the event that the next state is $y$, the time until the transition from $x$ to $y$ occurs has probability distribution $F_{xy}(\cdot|a)$

One form of the SMDP is to find a policy the minimizes the expected infinite horizon discounted cost, the "value" for each state:

$$\mathcal{E}\left\{\int_0^\infty e^{-\beta t}\rho(x(t), a(t))dt\right\},$$

where $x(t)$ and $a(t)$ denote, respectively, the state and action at time $t$.

For a fixed policy $\pi$, the value of a given state $x$ must satisfy

$$V_\pi(x) = \sum_{y\in\mathcal{X}} P_{xy}(\pi(x))\int_0^\infty\int_0^t e^{-\beta s}\rho(x,\pi(x))dsdF_{xy}(t|\pi(x)) +$$
$$\sum_{y\in\mathcal{X}} P_{xy}(\pi(x))\int_0^\infty e^{-\beta t}V_\pi(y)dF_{xy}(t|\pi(x)). \tag{1}$$

Defining

$$R(x,y,a) = \int_0^\infty\int_0^t e^{-\beta s}\rho(x,\pi(x))dsdF_{xy}(t|\pi(x)),$$

the expected reward that will be received on transition from state $x$ to state $y$ on action $a$, and

$$\gamma(x,y,a) = \int_0^\infty e^{-\beta t}dF_{xy}(t|\pi(x)),$$

the expected discount factor to be applied to the value of state $y$ on transition from state $x$ on action $a$, it is clear that equation (1) is nearly identical to the value-function equation for discrete time Markov reward processes,

$$V_\pi(x) = R(x, \pi(x)) + \gamma \sum_{y \in \mathcal{X}} P_{xy}(\pi(x)) V_\pi(y), \qquad (2)$$

where $R(x, a) = \sum_{y \in \mathcal{X}} P_{xy}(a) R(x, y, a)$. If transition times are identically one for an SMDP, then a standard discrete-time MDP results.

Similarly, while the value function associated with an *optimal* policy for an MDP satisfies the Bellman optimality equation

$$V^*(x) = \max_{a \in \mathcal{A}} \left\{ R(x, a) + \gamma \sum_{y \in \mathcal{X}} P_{xy}(a) V^*(y) \right\}, \qquad (3)$$

the optimal value function for an SMDP satisfies the following version of the Bellman optimality equation:

$$V^*(x) = \max_{a \in \mathcal{A}} \left\{ \sum_{y \in \mathcal{X}} P_{xy}(a) \int_0^\infty \int_0^t e^{-\beta s} \rho(x, a) ds dF_{xy}(t|a) + \right.$$

$$\left. \sum_{y \in \mathcal{X}} P_{xy}(a) \int_0^\infty e^{-\beta t} V^*(y) dF_{xy}(t|a) \right\}. \qquad (4)$$

## 3 Temporal Difference learning for SMDP's

Sutton's TD(0) [10] is a stochastic approximation method for finding solutions to the system of equations (2). Having observed a transition from state $x$ to state $y$ with sample reward $r(x, y, \pi(x))$, TD(0) updates the value function estimate $V^{(k)}(x)$ in the direction of the sample value $r(x, y, \pi(x)) + \gamma V^{(k)}(y)$. The TD(0) update rule for MDP's is

$$V^{(k+1)}(x) = V^{(k)}(x) + \alpha_k [r(x, y, \pi(x)) + \gamma V^{(k)}(y) - V^{(k)}(x)], \qquad (5)$$

where $\alpha_k$ is the learning rate. The sequence of value-function estimates generated by the TD(0) proceedure will converge to the true solution, $V_\pi$, with probability one [5,8,11] under the appropriate conditions on the $\alpha_k$ and on the definition of the MDP.

The TD(0) learning rule for SMDP's, intended to solve the system of equations (1) given a sequence of sampled state transitions, is:

$$V^{(k+1)}(x) = V^{(k)}(x) + \alpha_k \left[ \frac{1 - e^{-\beta\tau}}{\beta} r(x, y, \pi(x)) + e^{-\beta\tau} V^{(k)}(y) - V^{(k)}(x) \right], \qquad (6)$$

where the sampled transition time from state $x$ to state $y$ was $\tau$ time units, $\frac{1 - e^{-\beta\tau}}{\beta} r(x, y, \pi(x))$ is the sample reward received in $\tau$ time units, and $e^{-\beta\tau}$ is the sample discount on the value of the next state given a transition time of $\tau$ time units. The TD($\lambda$) learning rule for SMDP's is straightforward to define from here.

## 4   $Q$-learning for SMDP's

Denardo [6] and Watkins [12] define $Q_\pi$, the $Q$-function corresponding to the policy $\pi$, as

$$Q_\pi(x,a) = R(x,a) + \gamma \sum_{y \in \mathcal{X}} P_{xy}(a) V_\pi(y) \tag{7}$$

Notice that $a$ can be *any* action. It is not necesarily the action $\pi(x)$ that would be chosen by policy $\pi$. The function $Q^*$ corresponds to the optimal policy. $Q_\pi(x,a)$ represents the total discounted return that can be expected if any action is taken from state $x$, and policy $\pi$ is followed thereafter. Equation (7) can be rewritten as

$$Q_\pi(x,a) = R(x,a) + \gamma \sum_{y \in \mathcal{X}} P_{xy}(a) Q_\pi(y, \pi(y)), \tag{8}$$

and $Q^*$ satisfies the Bellman-style optimality equation

$$Q^*(x,a) = R(x,a) + \gamma \sum_{y \in \mathcal{X}} P_{xy}(a) \max_{a' \in \mathcal{A}} Q^*(y, a'), \tag{9}$$

$Q$-learning, first described by Watkins [12], uses stochastic approximation to iteratively refine an estimate for the function $Q^*$. The $Q$-learning rule is very similar to TD(0). Upon a sampled transition from state $x$ to state $y$ upon selection of $a$, with sampled reward $r(x,y,a)$, the $Q$-function estimate is updated according to

$$Q^{(k+1)}(x,a) = Q^{(k)}(x,a) + \alpha_k \left[ r(x,y,a) + \gamma \max_{a'} Q^{(k)}(y,a') - Q^{(k)}(x,a) \right]. \tag{10}$$

$Q$-functions may also be defined for SMDP's. The optimal $Q$-function for an SMDP satisfies the equation

$$\begin{aligned}
Q^*(x,a) = \; & \sum_{y \in \mathcal{X}} P_{xy}(a) \int_0^\infty \int_0^t e^{-\beta s} \rho(x,a) \, ds \, dF_{xy}(t|a) + \\
& \sum_{y \in \mathcal{X}} P_{xy}(a) \int_0^\infty e^{-\beta t} \max_{a' \in \mathcal{A}} Q^*(y,a') dF_{xy}(t|a).
\end{aligned} \tag{11}$$

This leads to the following $Q$-learning rule for SMDP's:

$$Q^{(k+1)}(x,a) = Q^{(k)}(x,a) + \alpha_k \left[ \frac{1 - e^{-\beta \tau}}{\beta} r(x,y,a) + e^{-\beta \tau} \max_{a'} Q^{(k)}(y,a') - Q^{(k)}(x,a) \right] \tag{12}$$

## 5   RTDP and Adaptive RTDP for SMDP's

The TD(0) and $Q$-learning algorithms are model-free, and rely upon stochastic approximation for asymptotic convergence to the desired function ($V_\pi$ and $Q^*$, respectively). Convergence is typically rather slow. Real-Time Dynamic Programming (RTDP) and Adaptive RTDP [1,3] use a system model to speed convergence.

RTDP assumes that a system model is known *a priori*; Adaptive RTDP builds a model as it interacts with the system. As discussed by Barto *et al.* [1], these asynchronous DP algorithms can have computational advantages over traditional DP algorithms even when a system model is given.

Inspecting equation (4), we see that the model needed by RTDP in the SMDP domain consists of three parts:

1. the state transition probabilities $P_{xy}(a)$,

2. the expected reward on transition from state $x$ to state $y$ using action $a$, $R(x, y, a)$, and

3. the expected discount factor to be applied to the value of the next state on transition from state $x$ to state $y$ using action $a$, $\gamma(x, y, a)$.

If the process dynamics are governed by a continuous time Markov chain, then the model needed by RTDP can be analytically derived through *uniformization* [2]. In general, however, the model can be very difficult to analytically derive. In these cases Adaptive RTDP can be used to incrementally build a system model through direct interaction with the system. One version of the Adaptive RTDP algorithm for SMDP's is described in Figure 1.

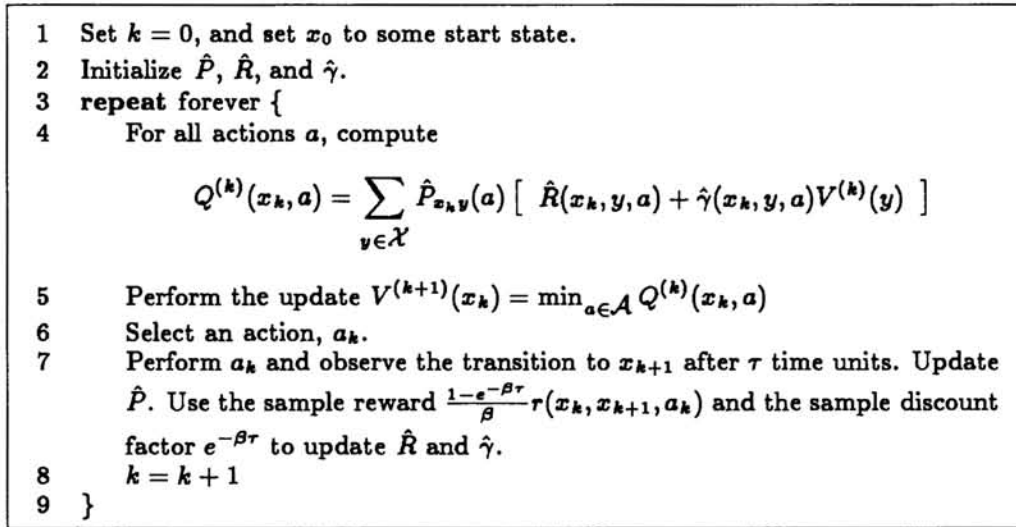

Figure 1: Adaptive RTDP for SMDP's. $\hat{P}$, $\hat{R}$, and $\hat{\gamma}$ are the estimates maintained by Adaptive RTDP of $P$, $R$, and $\gamma$.

Notice that the action selection procedure (line 6) is left unspecified. Unlike RTDP, Adaptive RTDP can not always choose the greedy action. This is because it only has an *estimate* of the system model on which to base its decisions, and the estimate could initially be quite inaccurate. Adaptive RTDP needs to explore, to choose actions that do not currently appear to be optimal, in order to ensure that the estimated model converges to the true model over time.

# 6  Experiment: Routing to two heterogeneous servers

Consider the queueing system shown in Figure 2. Arrivals are assumed to be Poisson with rate $\lambda$. Upon arrival, a customer must be routed to one of the two queues, whose servers have service times that are exponentially distributed with parameters $\mu_1$ and $\mu_2$ respectively. The goal is compute a policy that minimizes the objective function:

$$\mathcal{E}\left\{\int_0^\infty e^{-\beta t}[c_1 n_1(t) + c_2 n_2(t)]dt\right\},$$

where $c_1$ and $c_2$ are scalar cost factors, and $n_1(t)$ and $n_2(t)$ denote the number of customers in the respective queues at time $t$. The pair $(n_1(t), n_2(t))$ is the state of the system at time $t$; the state space for this problem is countably infinite. There are two actions available at every state: if an arrival occurs, route it to queue 1 or route it to queue 2.

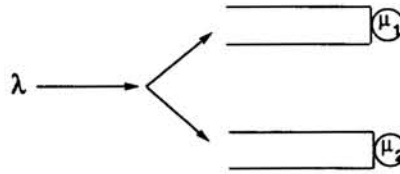

Figure 2: Routing to two queueing systems.

It is known for this problem (and many like it [7]), that the optimal policy is a *threshold* policy; *i.e.*, the set of states $S_1$ for which it is optimal to route to the first queue is characterized by a monotonically nondecreasing threshold function $F$ via $S_1 = \{(n_1, n_2) | n_1 \leq F(n_2)\}$. For the case where $c_1 = c_2 = 1$ and $\mu_1 = \mu_2$, the policy is simply to join the shortest queue, and the theshold function is a line slicing diagonally through the state space.

We applied the SMDP version of $Q$-learning to this problem in an attempt to find the optimal policy for some subset of the state space. The system parameters were set to $\lambda = \mu_1 = \mu_2 = 1$, $\beta = 0.1$, and $c_1 = c_2 = 1$. We used a feedforward neural network trained using backpropagation as a function approximator.

$Q$-learning must take exploratory actions in order to adequately sample all of the available state transitions. At each decision time $k$, we selected the action $a_k$ to be applied to state $x_k$ via the Boltzmann distribution

$$Pr\{a_k = a\} = \frac{e^{-Q^{(k)}(x_k,a)/T_k}}{\sum_{a' \in \mathcal{A}} e^{-Q^{(k)}(x_k,a')/T_k}},$$

where $T_k$ is the "computational temperature." The temperature is initialized to a relatively high value, resulting in a uniform distribution for prospective actions. $T_k$ is gradually lowered as computation proceeds, raising the probability of selecting actions with lower (and for this application, better) $Q$-values. In the limit, the action that is greedy with respect to the $Q$-function estimate is selected. The temperature and the learning rate $\alpha_k$ are decreased over time using a "search then converge" method [4].

Figure 3 shows the results obtained by $Q$-learning for this problem. Each square denotes a state visited, with $n_1(t)$ running along the $x$-axis, and $n_2(t)$ along the $y$-axis. The color of each square represents the probability of choosing action 1 (route arrivals to queue 1). Black represents probability 1, white represents probability 0. An optimal policy would be black above the diagonal, white below the diagonal, and could have arbitrary colors along the diagonal.

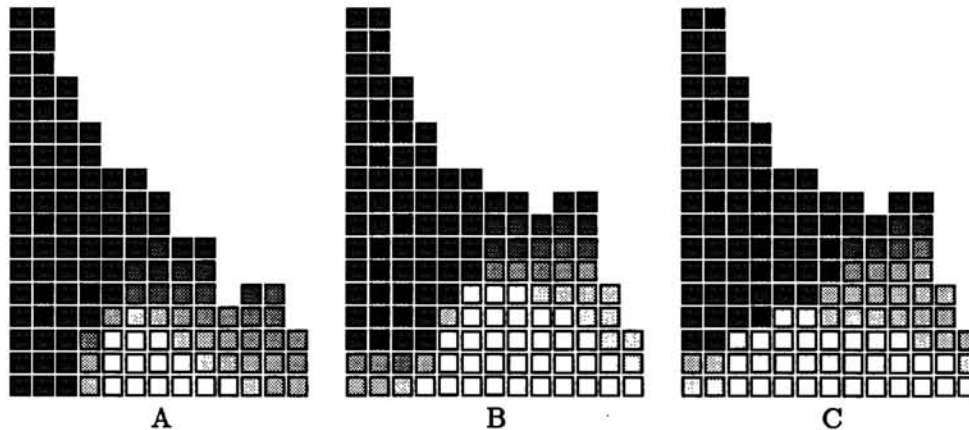

Figure 3: Results of the $Q$-learning experiment. Panel A represents the policy after 50,000 total updates, Panel B represents the policy after 100,000 total updates, and Panel C represents the policy after 150,000 total updates.

One unsatisfactory feature of the algorithm's performance is that convergence is rather slow, though the schedules governing the decrease of Boltzmann temperature $T_k$ and learning rate $\alpha_k$ involve design parameters whose tweakings may result in faster convergence. If it is known that the optimal policies are of theshold type, or that some other structural property holds, then it may be of extreme practical utility to make use of this fact by constraining the value-functions in some way or perhaps by representing them as a combination of appropriate basis vectors that implicity realize or enforce the given structural property.

## 7 Discussion

In this paper we have proposed extending the applicability of well-known reinforcement learning methods developed for discrete-time MDP's to the continuous time domain. We derived semi-Markov versions of TD(0), $Q$-learning, RTDP, and Adaptive RTDP in a straightforward way from their discrete-time analogues. While we have not given any convergence proofs for these new algorithms, such proofs should not be difficult to obtain if we limit ourselves to problems with finite state spaces. (Proof of convergence for these new algorithms is complicated by the fact that, in general, the state spaces involved are infinite; convergence proofs for traditional reinforcement learning methods assume the state space is finite.) Ongoing work is directed toward applying these techniques to more complicated systems, examining distributed control issues, and investigating methods for incorporating prior

knowledge (such as structured function approximators).

## Acknowledgements

Thanks to Professor Andrew Barto, Bob Crites, and to the members of the Adaptive Networks Laboratory. This work was supported by the National Science Foundation under Grant ECS-9214866 to Professor Barto.

# References

[1] A. G. Barto, S. J. Bradtke, and S. P. Singh. Learning to act using real-time dynamic programming. *Artificial Intelligence*. Accepted.

[2] D. P. Bertsekas. *Dynamic Programming: Deterministic and Stochastic Models*. Prentice Hall, Englewood Cliffs, NJ, 1987.

[3] S. J. Bradtke. *Incremental Dynamic Programming for On-line Adaptive Optimal Control*. PhD thesis, University of Massachusetts, 1994.

[4] C. Darken, J. Chang, and J. Moody. Learning rate schedules for faster stochastic gradient search. In *Neural Networks for Signal Processing 2 — Proceedings of the 1992 IEEE Workshop*. IEEE Press, 1992.

[5] P. Dayan and T. J. Sejnowski. Td($\lambda$): Convergence with probability 1. *Machine Learning*, 1994.

[6] E. V. Denardo. Contraction mappings in the theory underlying dynamic programming. *SIAM Review*, 9(2):165–177, April 1967.

[7] B. Hajek. Optimal control of two interacting service stations. *IEEE-TAC*, 29:491–499, 1984.

[8] T. Jaakkola, M. I. Jordan, and S. P. Singh. On the convergence of stochastic iterative dynamic programming algorithms. *Neural Computation*, 1994.

[9] S. M. Ross. *Applied Probability Models with Optimization Applications*. Holden-Day, San Francisco, 1970.

[10] R. S. Sutton. Learning to predict by the method of temporal differences. *Machine Learning*, 3:9–44, 1988.

[11] J. N. Tsitsiklis. Asynchronous stochastic approximation and Q-learning. Technical Report LIDS-P-2172, Laboratory for Information and Decision Systems, MIT, Cambridge, MA, 1993.

[12] C. J. C. H. Watkins. *Learning from Delayed Rewards*. PhD thesis, Cambridge University, Cambridge, England, 1989.